# Dimensionality Reduction and Prior Knowledge in E-set Recognition

**Kevin J. Lang**[1]
Computer Science Dept.
Carnegie Mellon University
Pittsburgh, PA 15213
USA

**Geoffrey E. Hinton**
Computer Science Dept.
University of Toronto
Toronto, Ontario M5S 1A4
Canada

## ABSTRACT

It is well known that when an automatic learning algorithm is applied to a fixed corpus of data, the size of the corpus places an upper bound on the number of degrees of freedom that the model can contain if it is to generalize well. Because the amount of hardware in a neural network typically increases with the dimensionality of its inputs, it can be challenging to build a high-performance network for classifying large input patterns. In this paper, several techniques for addressing this problem are discussed in the context of an isolated word recognition task.

## 1  Introduction

The domain for our research was a speech recognition task that requires distinctions to be learned between recordings of four highly confusable words: the names of the letters "B", "D", "E", and "V". The task was created at IBM's T. J. Watson Research Center, and is difficult because many speakers were included and also because the recordings were made under noisy office conditions using a remote microphone. One hundred male speakers said each of the 4 words twice, once for training and again for testing. The words were spoken in isolation, and the recordings averaged 1.1 seconds in length. The signal-to-noise ratio of the data set has been estimated to be about 15 decibels, as compared to

50 decibels for typical lip-mike recordings (Brown, 1987). The key feature of the data set from our point of view is that each utterance contains a tiny information-laden event — the release of the consonant — which can easily be overpowered by meaningless variation in the strong "E" vowel and by background noise.

Our first step in processing these recordings was to convert them into spectrograms using a standard DFT program. The spectrograms encoded the energy in 128 frequency bands (ranging up to 8 kHz) at 3 msec intervals, and so they contained an average of about 45,000 energy values. Thus, a naive back-propagation network which devoted a separate weight to each of these input components would contain far too many weights to be properly constrained by the task's 400 training patterns.

As described in the next section, we drastically reduced the dimensionality of our training patterns by decreasing their resolution in both frequency and time and also by using a segmentation algorithm to extract the most relevant portion of each pattern. However, our network still contained too many weights, and many of them were devoted to detecting spurious features. This situation motivated the experiments with our network's objective function and architecture that will be described in sections 3 and 4.

## 2    Reducing the Dimensionality of the Input Patterns

Because it would have been futile to feed our gigantic raw spectrograms into a back-propagation network, we first decreased the time resolution of our input format by a factor of 4 and the frequency resolution of the format by a factor 8. While our compression along the time axis preserved the linearity of the scale, we combined different numbers of raw freqencies into the various frequency bands to create a mel scale, which is linear up to 2 kHz and logarithmic above that, and thus provides more resolution in the more informative lower frequency bands.

Next, a segmentation heuristic was used to locate the consonant in each training pattern so that the rest of the pattern could be discarded. On average, all but 1/7 of each recording was thrown away, but we would have liked to have discarded more. The useful information in a word from the E-set is concentrated in a roughly 50 msec region around the consonant release in the word, but current segmentation algorithms aren't good enough to accurately position a 50 msec window on that region. To prevent the loss of potentially useful information, we extracted a 150 msec window from around each consonant release. This safeguard meant that our networks contained about 3 times as many weights as would be required with an ideal segmentation.

We were also concerned that segmentation errors during recognition could lower our final system's performance, so we adopted a simple segmentation-free testing method in which the trained network is scanned over the full-length version of each testing utterance. Figures 3(a) and 3(b) show the activation traces generated by two different networks when scanned over four sample utterances. To the right of each of the capital letters which identifies a particular sample word is a set of 4 wiggly lines that should be viewed as the output of a 4-channel chart recorder which is connected to the network's four output units. Our recognition rule for unsegmented utterances states that the output unit which

## output unit weights

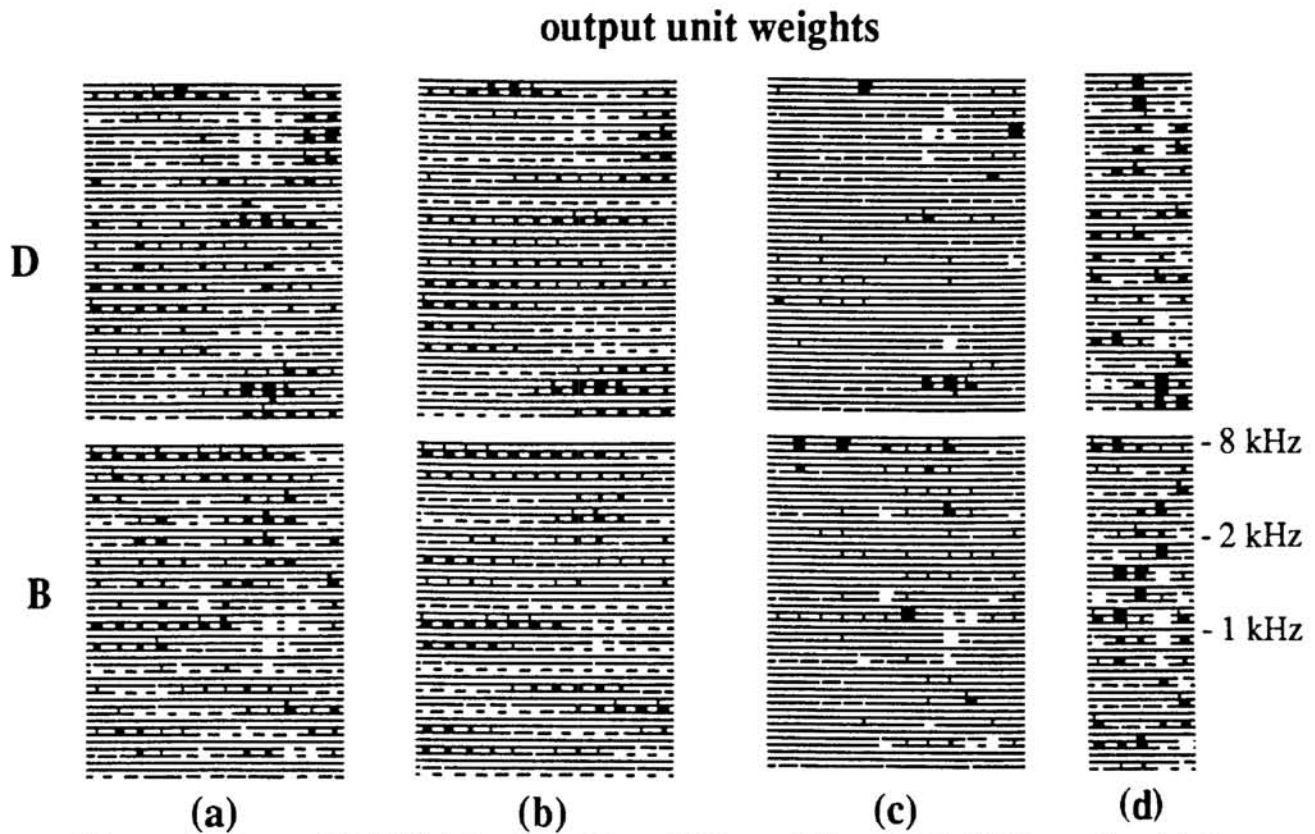

**Figure 1:** Output Unit Weights from Four Different 2-layer BDEV Networks: (a) baseline, (b) smoothed, (c) decayed, (d) TDNN

generates the largest activation spike (and hence the highest peak in the chart recorder's traces) on a given utterance determines the network's classification of that utterance.[2]

To establish a performance baseline for the experiments that will be described in the next two sections, we trained the simple 2-layer network of figure 2(a) until it had learned to correctly identify 94 percent of our training segments.[3]

This network contains 4 output units (one for each word) but no hidden units.[4] The weights that this network used to recognize the words **B** and **D** are shown in figure 1(a). While these weight patterns are quite noisy, people who know how to read spectrograms can see sensible feature detectors amidst the clutter. For example, both of the units appear to be stimulated by an energy burst near the 9th time frame. However, the units expect to see this energy at different frequencies because the tongue position is different in the consonants that the two units represent.

Unfortunately, our baseline network's weights also contain many details that don't make

any sense to speech recognition experts. These spurious features are artifacts of our small, noisy training set, and are partially to blame for the very poor performance of the network; it achieved only 37 percent recognition accuracy when scanned across the unsegmented testing utterances.

## 3  Limiting the Complexity of a Network using a Cost Function

Our baseline network performed poorly because it had lots of free parameters with which it could model spurious features of the training set. However, we had already taken our brute force techniques for input dimensionality reduction (pre-segmenting the utterances and reducing the resolution of input format) about as far as possible while still retaining most of the useful information in the patterns. Therefore it was necessary to resort to a more subtle form of dimensionality reduction in which the back-propagation learning algorithm is allowed to create complicated weight patterns only to the extent that they actually reduce the network's error.

This constraint is implemented by including a cost term for the network's complexity in its objective function. The particular cost function that should be used is induced by a particular definition of what constitutes a complicated weight pattern, and this definition should be chosen with care. For example, the rash of tiny details in figure 1(a) originally led us to penalize weights that were different from their neighbors, thus encouraging the network to develop smooth, low-resolution weight patterns whenever possible.

$$C = \frac{1}{2} \sum_i \frac{1}{\|\mathcal{N}_i\|} \sum_{j \in \mathcal{N}_i} (w_i - w_j)^2 \tag{1}$$

To compute the total tax on non-smoothness, each weight $w_i$ was compared to all of its neighbors (which are indexed by the set $\mathcal{N}_i$). When a weight differed from a neighbor, a penalty was assessed that was proportional to the square of their difference. The term $\|\mathcal{N}_i\|^{-1}$ normalized for the fact that units at the edge of a receptive field have fewer neighbors than units in the middle.

When a cost function is used, a tradeoff factor $\lambda$ is typically used to control the relative importance of the error and cost components of the overall objective function $O = E + \lambda C$. The gradient of the overall objective function is then $\nabla O = \nabla E + \lambda \nabla C$. To compute $\nabla C$, we needed the derivative of our cost function with respect to each weight $w_i$. This derivative is just the difference between the weight and the average of its neighbors: $\frac{\partial C}{\partial w_i} = w_i - \frac{1}{\|\mathcal{N}_i\|} \sum_{j \in \mathcal{N}_i} w_j$, so minimizing the combined objective function was equivalent to minimizing the network's error while simultaneously smoothing the weight patterns by decaying each weight towards the average of its neighbors.

Figure 1(b) shows the **B** and **D** weight patterns of a 2-layer network that was trained under the influence of this cost function. As we had hoped, sharp transitions between neighboring weights occurred primarily in the maximally informative consonant release of each word, while the spurious details that had plagued our baseline network were smoothed out of existence. However, this network was even worse at the task of generalizing to unsegmented test cases than the baseline network, getting only 35 percent of

them correct.

While equation 1 might be a good cost function for some other task, it doesn't capture our prior knowledge that the discrimination cues in E-set recognition are highly localized in time. This cost function tells the network to treat unimportant neighboring input components similarly, but we really want to tell the network to ignore these components altogether. Therefore, a better cost function for this task is the one associated with standard weight decay:

$$C = \frac{1}{2} \sum_i w_i^2 \tag{2}$$

Equation 2 causes weights to remain close to zero unless they are particularly valuable for reducing the network's error on the training set. Unfortunately, the weights that our network learns under the influence of this function merely look like smaller versions of the baseline weights of figure 1(a) and perform just as poorly. No matter what value is used for $\lambda$, there is very little size differentiation between the weights that we know to be valuable for this task and the weights that we know to be spurious. Weight decay fails because our training set is so small that spurious weights do not appear to be as irrelevant as they really are for performing the task in general. Fortunately, there is a modified form of weight decay (Scalettar and Zee, 1988) that expresses the idea that the disparity between relevant and irrelevant weights is greater than can be deduced from the training set:

$$C = \frac{1}{2} \sum_i \frac{w_i^2}{2.5 + w_i^2} \tag{3}$$

The weights of figure 1(c) were learned under the influence of equation 3.[5] In these patterns, the feature detectors that make sense to speech recognition experts stand out clearly above a highly suppressed field of less important weights. This network generalizes to 48 percent of the unsegmented test cases, while our earlier networks had managed only 37 percent accuracy.

## 4    A Time-Delay Neural Network

The preceding experiments with cost functions show that controlling attention (rather than resolution) is the key to good performance on the **BDEV** task. The only way to accurately classify the utterances in this task is to focus on the tiny discrimination cues in the spectrograms while ignoring the remaining material in the patterns.

Because we know that the **BDEV** discrimination cues are highly localized in time, it would make sense to build a network whose architecture reflected that knowledge. One such network (see figure 2(b)) contains many copies of each output unit. These copies apply identical weight patterns to the input in all possible positions. The activation values

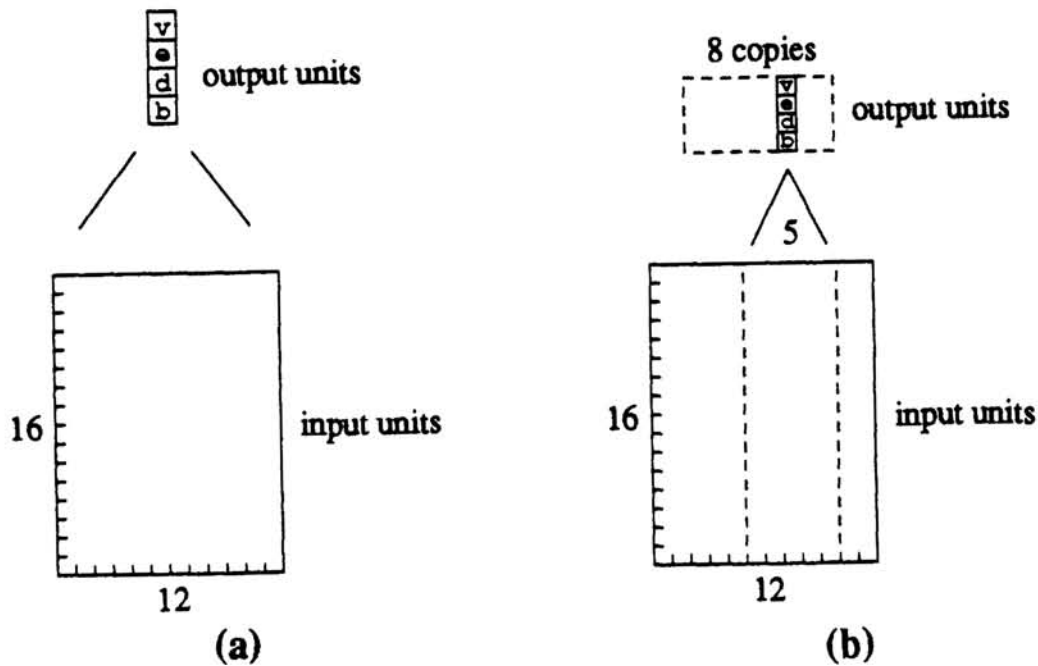

**Figure 2:** Conventional and Time-Delay 2-layer Networks

from all of the copies of a given output unit are summed to generate the overall output value for that unit.[6]

Now, assuming that the learning algorithm can construct weight patterns which recognize the characteristic features of each word while rejecting the rest of the material in the words, then when an instance of a particular word is shown to the network, the only unit that will be activated is the output unit copy for that word which happens to be aligned with the recognition cues in the pattern. Then, the summation step at the output stage of the network serves as an OR gate which transmits that activation to the outside world.

This network architecture, which has been named the "Time-Delay Neural Network" or "TDNN", has several useful properties for E-set recognition, all of which are consequences of the fact that the network essentially performs its own segmentation by recognizing the most relevant portion of each input and rejecting the rest. One benefit is that sharp weight patterns can be learned even when the training patterns have been sloppily segmented. For example, in the TDNN weight patterns of figure 1(d), the release-burst detectors are localized in a single time frame, while in the earlier weight patterns from conventional networks they were smeared over several time frames.

Also, the network learns to actively discriminate between the relevant and irrelevant portions of its training segments, rather than trying to ignore the latter by using small weights. This turns out to be a big advantage when the network is later scanned across unsegmented utterances, as evidenced by the vastly different appearances of the output

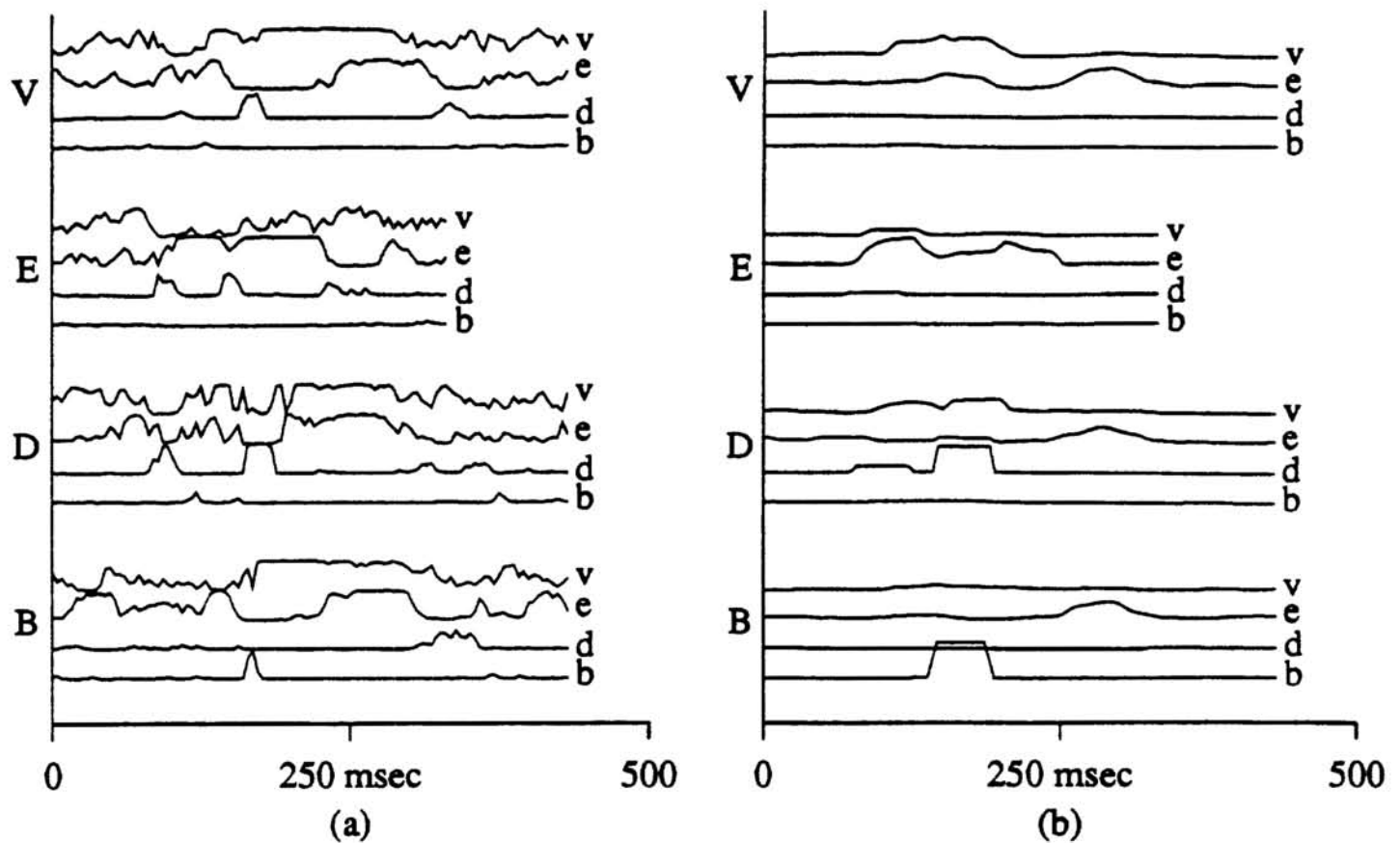

**Figure 3:** Output Unit Activation Traces of a Conventional Network and a Time-Delay Network, on Four Sample Utterances

activity traces in figures 3(a) and 3(b).[7]

Finally, because the TDNN can locate and attend to the most relevant portion of its input, we are able to make its receptive fields very narrow, thus reducing the number of free parameters in the network and making it highly trainable with the small number of training cases that are available in this task. In fact, the scanning mode generalization rate of our 2-layer TDNN is 65 percent, which is nearly twice the accuracy of our baseline 2-layer network.

## 5  Comparison with other systems

The 2-layer networks described up to this point were trained and tested under identical conditions so that their performances could be meaningfully compared. No attempt was made to achieve really high performance in these experiments. On the other hand when

we trained a 3-layer TDNN using the slightly fancier methodology described in (Lang, Hinton, and Waibel, 1990),[8] we obtained a system that generalized to about 91 percent of the unsegmented test cases. By comparison, the standard, large-vocabulary IBM hidden Markov model accounts for 80 percent of the test cases, and the accuracy of human listeners has been measured at 94 percent. In fact, the TDNN is probably the best automatic recognition system built for this task to date; it even performs slightly better than the continuous acoustic parameter, maximum mutual information hidden Markov model proposed in (Brown, 1987).

# 6 Conclusion

The performance of a neural network can be improved by building *a priori* knowledge into the network's architecture and objective function. In this paper, we have exhibited two successful examples of this technique in the context of a speech recognition task where the crucial information for making an output decision is highly localized and where the number of training cases is limited. Tony Zee's modified version of weight decay and our time-delay architecture both yielded networks that focused their attention on the short-duration discrimination cues in the utterances. Conversely, our attempts to use weight smoothing and standard weight decay during training got us nowhere because these cost functions didn't accurately express our knowledge about the task.

## Acknowledgements

This work was supported by Office of Naval Research contract N00014-86-K-0167, and by a grant from the Ontario Information Techology Research Center. Geoffrey Hinton is a fellow of the Canadian Institute for Advanced Research.

## Footnotes

[1]Now at NEC Research Institute, 4 Independence Way, Princeton, NJ 08540.

[2]One can't reasonably expect a network that has been trained on pre-segmented patterns to function well when tested in this way, but our best network (a 3-layer TDNN) actually does perform better in this mode than when trained and tested on segments selected by a Viterbi alignment with an IBM hidden Markov model. Moreover, because the Viterbi alignment procedure is told the identity of the words in advance, it is probably more accurate than any method that could be used in a real recognition system.

[3]This rather arbitrary halting rule for the learning procedure was uniformly employed during the experiments of sections 2, 3 and 4.

[4]Experiments performed with multi-layer networks support the same general conclusions as the results reported here.

[5]We trained with $\lambda = 100$ here as opposed to the setting of $\lambda = 10$ that worked best with standard weight decay.

[6]We actually designed this network before performing our experiments with cost functions, and were originally attracted by its translation invariance rather than by the advantages mentioned here (Lang, 1987).

[7]While the main text of this paper compares the performance of a sequence of 2-layer networks, the plots of figure 3 show the output traces of 3-layer versions of the networks. The correct plots could not be conveniently generated because our CMU Common Lisp program for creating them has died of bit rot.

[8]Wider but less precisely aligned training segments were employed, as well as randomly selected "counter-example" segments that further improved the network's already good "E" and background noise rejection. Also, a preliminary cross-validation run was performed to locate a nearly optimal stopping point for the learning procedure. When trained using this improved methodology, a conventional 3-layer network achieved a generalization score in the mid 50's.

## References

P. Brown. (1987) *The Acoustic-Modeling Problem in Automatic Speech Recognition.* Doctoral Dissertation, Carnegie Mellon University.

K. Lang. (1987) *Connectionist Speech Recognition.* PhD Thesis Proposal, Carnegie Mellon University.

K. Lang, G. Hinton, and A. Waibel. (1990) A Time-Delay Neural Network Architecture for Isolated Word Recognition. *Neural Networks* 3(1).

R. Scalettar and A. Zee. (1988) In D. Waltz and J. Feldman (eds.), *Connectionist Models and their Implications*, p. 309. Publisher: A. Blex.

